# A VLSI Neural Network for Color Constancy

**Andrew Moore**
Computation and Neural Systems Program, 116-81
California Institute of Technology
Pasadena, CA 91125

**Geoffrey Fox***
Dept. of Physics
California Institute of Technology
Pasadena, CA 91125

**John Allman**
Dept. of Biology, 216-76
California Institute of Technology
Pasadena, CA 91125

**Rodney Goodman**
Dept. of Electrical Engineering, 116-81
California Institute of Technology
Pasadena, CA 91125

## Abstract

A system for color correction has been designed, built, and tested successfully; the essential components are three custom chips built using sub-threshold analog CMOS VLSI. The system, based on Land's Retinex theory of color constancy, produces colors similar in many respects to those produced by the visual system. Resistive grids implemented in analog VLSI perform the smoothing operation central to the algorithm at video rates. With the electronic system, the strengths and weaknesses of the algorithm are explored.

## 1  A MODEL FOR COLOR CONSTANCY

Humans have the remarkable ability to perceive object colors as roughly constant even if the color of the illumination is varied widely. Edwin Land, founder of the Polaroid Corporation, models the computation that results in this ability as three identical center-surround operations performed independently in three color planes, such as red, green, and blue (Land, 1986). The basis for this model is as follows.

Consider first an array of grey papers with different reflectances. (Land designated these arrays *Mondrians*, since they resemble the works of the Dutch painter Piet

Mondrian.) Land illuminated a Mondrian with a gradient of illumination, ten times more bright at the top than at the bottom, so that the flux reaching the eye from a dark grey patch at top was identical to the flux from a light grey patch at bottom. Subjects reported that the top paper was dark grey and the bottom paper was light grey. Land accounted for this with a center minus surround model. At each point in an image, the incoming light is compared to a spatial average of light in the neighborhood of the point in question. Near the top of the Mondrian, the abundance of white is sensed and subtracted from the central sensor to normalize the central reading with respect to neighboring values, weighted with distance; near the bottom, the abundance of dark is sensed and used to correct the central reading. Land proposed that the weighting function of the surround is a monotonic decreasing function of distance, such as $1/r^2$.

In earlier work, similar experiments were carried out with color Mondrians (Land, 1977; McCann *et. al.*, 1976). However, instead of varying the intensity of illumination, Land and his colleagues varied the *color* of the illumination. The color of patches in a Mondrian remained nearly constant despite large changes in the illuminant color. This is the phenomenon of color constancy: the ability of observers to judge, under a wide variety of lighting conditions, the approximate reflectance or intrinsic color of objects. Land and his colleagues proposed a variety of different models for this phenomenon, collectively referred to as *Retinex* models. (The term Retinex was coined by Land since he was not sure whether the computation was going on in the retina, the cortex, or both.) In his most recent paper on the subject (Land, 1986), Land simply extended the black-and-white model to the three color dimensions. In each of three independent color planes, the color at a given point is compared to that of the points surrounding it, weighted as $1/r^2$.

## 2    EFFICIENT CALCULATION OF THE SURROUND

In practical terms, the Retinex algorithm corresponds to subtracting from an image a blurred version of itself. The distance weighting (type of blurring) Land proposes varies as $1/r^2$, so the operation is a center minus surround operation, where the surround is the center convolved with a $1/r^2$ kernel.

$$l'_{out,i}(x,y) = l'_i(x,y) - \log\left[l_i(x,y) \otimes \frac{1}{r^2}\right] \qquad r \neq 0 \qquad (1)$$

where $l_i$ is the signal or lightness in color plane $i$, and $l'_i$ is the log of the signal. The logs are important since the signal is composed of illuminant times reflectance and the log of a product is a sum. By subtracting the blurred version of the image after taking logs, the illuminant is subtracted away in the ideal case (but see below).

This type of Retinex algorithm, then, has a psychophysical basis and sound computational underpinnings (Hurlbert, 1986). But the complexity is too great. Since the required surround is so large, such a convolution across an $N x N$ pixel image entails on the order of $N^4$ operations. On a chip, this corresponds to explicit connections from each pixel to most if not all other pixels.

A similar operation can be carried out much more efficiently by switching from

a convolution to a resistive grid calculation. The operations are similar since the weighting of neighboring points (Green's function) in a resistive grid decreases in the limit as the exponential of the distance from a given location on a resistive grid (Mead, 1989). Again, the kernel is a monotonic decreasing function. With this type of kernel, the operation in each Retinex (color channel) is

$$l'_{out,i}(x,y) = l'_i(x,y) - l'_i(x,y) \otimes e^{-\frac{|r|}{\lambda}} \tag{2}$$

where $\lambda$ is the length constant or extent of weighting in the grid. Since the calculation is purely local, the complexity is reduced dramatically from $O(N^4)$ to $O(N^2)$. On a chip, a local computation corresponds to connections only between nearest-neighbor pixels.

# 3    EVALUATION OF THE ALGORITHM WITH COMPUTER SIMULATIONS

## 3.1   STRENGTHS AND WEAKNESSES OF THE ALGORITHM

Images of a subject holding a color poster were captured under fluorescent and incandescent light with an RGB video camera and a 24 bit frame grabber. First, the camera was adjusted so that the color looked good under fluorescent light. Next, without readjusting the camera, the fluorescents were turned off and the subject was illuminated with incandescent light. The results were unacceptable. The skin color was very red, and, since the incandescent lamp was not very bright, the background was lost in darkness. The two images were processed with the Land algorithm, using resistive grids to form the surround for subtraction. Details of the simulations and color images can be found in (Moore et. al, 1991). For the good, fluorescent image, the processing improved the image contrast somewhat. For the poor, incandescent image, the improvement was striking. Skin color was nearly normal, shadows were softened, and the the background was pulled out of darkness.

Computer simulation also pointed out two weaknesses of the algorithm: color Mach bands and the greying out of large monochromatic regions. Color Mach bands arise from this algorithm in the following way. Suppose that a strongly colored region, e.g. red, abuts a grey region. In the grey region, the surround subtracted at a given point has a strong red component. Therefore, after subtraction of the surround, a grey point is rendered as grey minus red, or equivalently, grey plus the complementary color of red, which is blue-green. Since the surround weighting decreases with distance, the points in the image closest to the red area are strongly tinged with blue-green, while points further away are less discolored. Induction of this sort in black-and-white images is known as the Mach band effect. An analogous induction effect in color is intrinsic to this algorithm.

Greying out of large colored areas is also an intrinsic weakness of the algorithm. The surrounds used in the simulations are quite large, with a length constant of nearly one third of the image. Often a large portion of an image is of a single color, e.g. a blue sky commonly fills the upper half of many natural scenes. In the sky region, the surround samples mostly blue, and with subtraction, blue is subtracted from blue, leaving a grey sky. This effect illustrates the essence of the algorithm

- it operates under a *grey world assumption*. The image for which this algorithm is ideal is richly colored, with reds and their green complements, yellows and their blue complements, and whites with their black complements. In such images, the large surround is sampling the color of a grey "mirror", since the sum of a color and its complement is grey. If this condition holds, the color subtracted when the surround is subtracted from a point in the image is the color of the illuminant; the surround acts as a dull grey mirror which reflects the illuminant. [Many color constancy schemes rely on this assumption; for a review see (Lennie and D'Zmura, 1988).]

## 3.2   AN EXTENSION TO THE LAND ALGORITHM

These two weaknesses arise from too much surround subtraction in solidly colored areas. One way the minimize the effects is to modulate the surround with a measure of image structure, which we call *edginess*, before subtraction. So, while for the original algorithm, the operation is *output = center − surround* , to ameliorate induction effects and lessen reliance on the grey world assumption, the surround weight should be modified pointwise. In particular, if edginess is given a value close to zero in homogeneous regions like the blue sky, and is given a value close to one in detailed areas, a better formulation is *output = center − surround · edginess*. In this relation, the surround is effectively zeroed in smooth areas before it is subtracted, so that induction is diminished - more of the original color is retained. The extended algorithm, then, is a working compromise between color constancy via strict application of the grey world assumption and no color constancy at all. To compute a measure of spatial structure, the average magnitude of the first spatial derivatives is found at each point in each color plane is smoothed on a resistive grid; the output at a given point is multiplied with the surround value from the corresponding point of first resistive grid. In our simulations, the modified algorithm reduces (but does not eliminate) color Mach bands, and returns color to large monochromatic regions such as the the sky in the example image discussed above, at the cost of one additional resistive grid per color channel. This extension is not the whole answer, however. If a large region is highly textured (for example, if there is a flock of birds in the sky), edginess is high, the surround is subtracted at near full strength, and the sky is rendered grey in the textured region. This is a subject of continuing research. We implemented the original algorithm, but not this extension of it, using analog VLSI.

## 4   VLSI IMPLEMENTATION OF THE RETINEX ALGORITHM

To realize a real-time electronic system of video camera color correction based on Land's algorithm, the three color outputs of a video camera are fed onto three separate resistive grids built from subthreshold analog CMOS VLSI. Each 48 by 47 node resistive grid was built using 2 micron design rules and contains about 60,000 transistors. The circuit details within each pixel are similar to those of the analog retina (Mead, 1989); technical details of the system may be found in (Moore *et.al.*, 1991).

Computer simulations are quite costly in terms of time and disk storage. With a real-time system, it is possible to intensively investigate the strengths and weaknesses of this color correction algorithm quickly and economically.

## 4.1   REAL-TIME VERIFICATION OF ALGORITHM STRENGTHS

### 4.1.1   Dynamic range enhancement

A common problem with video imaging is that the range of an image exceeds the dynamic range of the camera sensors. For example, consider an image comprised of an indoor scene and an outdoor scene viewed through a window. The indoor illumination (e.g., direct sunlight) can be one thousand times or more brighter than the indoor illumination (e.g., artificial lights or indirect sunlight). A video camera can only capture one portion of the scene with fidelity. By opening up the camera iris so that a lot of light falls on the camera sensors, the indoor scene looks good, but the outdoor scene is awash in white. Conversely, by closing the camera iris so that less light falls on the camera sensors, the outdoor scene looks good, but the indoor scene is rendered as deep black.

In fact, the image information is often not lost in this troublesome situation. Most sensors are not linear, but instead have a response function that resembles a hyperbolic tangent. Rather than saturating at the extremes of the response range, most sensors compress information near those response extremes. With a center-surround processing stage following a camera, the information "squashed" near the camera range limits can be recovered. In extremely bright portions of an image, white is subtracted from white, "pulling" the signal toward the mid-range, so that details in that portion of the scene become defined. Similarly, in dark portions of the scene, dark is subtracted from dark and the details of the indoor portion of the example image are visible. Thus the Land algorithm as applied to video imaging can enhance the dynamic range of video cameras. [This strength of the algorithm was predicted from the similar capability of the (black-and-white) silicon retina (Mead, 1989) - it has a dynamic range that exceeds by far the range of conventional cameras since it incorporates light sensors and center-surround processing on one chip.]

### 4.1.2   Color constancy

For a richly colored scene, the Land algorithm can remove strongly colored illumination, with some qualifications. We constructed a color Mondrian with many differently colored patches of paper, and illuminated it with ordinary fluorescent light plus various colored lights. Under a wide range of conditions, the color of the Mondrian as viewed on a video monitor changes with the illumination while it looks fairly stable to an observer. After passing the images through the electronic color compensation system, the image is also fairly stable for a wide variety of illumination conditions. There is a significant difference, however, between what an observer sees and what the corrected camera image reports. The video images passed through the electronic implementation of the Land algorithm take on the illuminant somewhat in portions of the image that are brighter than average, and take on the complementary color of the illuminant in portions that are darker than average. For example, for a blue illumination, the raw video image looks bluer all over. The processed image changes in a different way. White patches are faintly

blue (much less as compared to the raw image), and black patches (which remain black in the raw image) are tinged with yellow. There is psychophysical evidence that the same effects are noted by human observers (see Jameson and Hurvich, 1989, for a review), but they are much less pronounced than those produced by the Land algorithm in our experience. Still, the overall effect of constancy in the processed images is convincing as compared to the raw images.

## 4.2   REAL-TIME VERIFICATION OF ALGORITHM WEAKNESSES

### 4.2.1   Color Mach bands and greying of large regions

To our surprise, the color Mach band effect, explained above, is less pronounced than we expected; for many scenes the induction effects are not noticeable. It is possible to the see the Mach bands clearly by placing colored cards on a grey background - the complementary color of the card surrounds the card as a halo that diminishes with distance from the card.

Since the Retinex algorithm relies on the grey world assumption, the algorithm fails where this assumption fails to hold. With the real-time system, we have demonstrated this in many ways. For example, if the video camera is pointed at the color Mondrian and the hand of a Caucasian investigator (with a reddish skin tone) is slowly moved in front of the camera lens, the Mondrian in the background slowly grows more green. Green is the complementary color of red. Another example of practical importance is revealed by zooming in on a particular patch of the Mondrian. As more and more of the image is filled with this patch, the patch grows greyer and greyer, because the correction system subtracts the patch color from itself.

### 4.2.2   Scene dependence of color constancy

As described above, we were impressed with this algorithm after simulating it on a digital computer. The skin tone of a subject, deeply reddened by incandescent light, was dramatically improved by the algorithm. In the computer study, the subject's face was, by accident rather than design, just in the middle of a large white patch and a large black patch. The electronic system yields perfect constancy of skin tone with this configuration also, but not for an arbitrary configuration. In short, the color constancy afforded by this algorithm is scene dependent; to consistently produce perfect color constancy of an object with the real-time system, it is necessary to place the object carefully within a scene. We are still investigating this weakness of the algorithm. Whether it is camera dependent (i.e., the result of camera nonlinearities) remains to be seen.

## 5   Conclusion

After studying the psychophysics and the computational issues in color constancy, encouraging preliminary results for a particular version of Land's Retinex algorithm were obtained in computer simulation. In order to study the algorithm intensively, an electronic system was developed; the system uses three resistive grids built from

subthreshold analog CMOS VLSI to form a blurred version of the image for subtraction from the original. It was found that the system produces images that are more constant, in a sense, than raw video images when the illuminant color varies. However, the constancy is more apparent than real; if absolute constancy of a particular object is desired, that object must be carefully placed in its surroundings. The real-time system allowed us to address this and other such practical issues of the algorithm for the first time.

## Acknowledgements

We are grateful to many of our colleagues at Caltech and elsewhere for discussions and support in this endeavor. A.M. was supported by fellowships from the Parsons Foundation and the Pew Charitable Trust and by research assistantships from Office of Naval Research, the Joint Tactical Fusion Program and the Center for Research in Parallel Computation. We are grateful to DARPA for MOSIS fabrication services, and to Hewlett Packard for computing support in the Mead Lab. The California Institute of Technology has filed for a U.S. patent for this and other related work.

## Footnotes

*Present address: Dept. of Physics, Syracuse University, Syracuse, NY 13244

## References

A. Hurlbert. (1986) Formal connections between lightness algorithms. *J. Opt. Soc. Am.* **A3**: 1684-1693.

D. Jameson & L.M. Hurvich (1989). Essay concerning color constancy. *Ann. Rev. Psychol.* **40**:1-22.

E.H. Land. (1977) The Retinex theory of color vision. *Scientific American* 237:108-128.

E.H. Land. (1986) An alternative technique for the computation of the designator in the retinex theory of color vision. *Proc. Natl. Acad. Sci. USA* **83**:3078-3080.

P. Lennie & M. D'Zmura. (1988) Mechanisms of color vision. *CRC Crit. Rev. Neurobiol.* **3**(4):333-400.

J.J. McCann, S.P. McKee, & T.H. Taylor. (1976) Quantitative studies in Retinex theory. *Vision Res.* **16**:445-458.

C.A. Mead. (1989) *Analog VLSI and Neural Systems*. Reading, MA: Addison-Wesley.

A. Moore, J. Allman, & R. Goodman. (1991) A Real-time Neural System for Color Constancy. *IEEE Trans. Neural Networks* **2**(2) *In press*.